# "Name That Song!": A Probabilistic Approach to Querying on Music and Text

**Eric Brochu**
Department of Computer Science
University of British Columbia
Vancouver, BC, Canada
ebrochu@cs.ubc.ca

**Nando de Freitas**
Department of Computer Science
University of British Columbia
Vancouver, BC, Canada
nando@cs.ubc.ca

## Abstract

We present a novel, flexible statistical approach for modelling music and text jointly. The approach is based on multi-modal mixture models and *maximum a posteriori* estimation using EM. The learned models can be used to browse databases with documents containing music and text, to search for music using queries consisting of music and text (lyrics and other contextual information), to annotate text documents with music, and to automatically recommend or identify similar songs.

## 1 Introduction

Variations on "name that song"-types of games are popular on radio programs. DJs play a short excerpt from a song and listeners phone in to guess the name of the song. Of course, callers often get it right when DJs provide extra contextual clues (such as lyrics, or a piece of trivia about the song or band). We are attempting to reproduce this ability in the context of information retrieval (IR). In this paper, we present a method for querying with words and/or music.

We focus on monophonic and polyphonic musical pieces of known structure (MIDI files, full music notation, etc.). Retrieving these pieces in multimedia databases, such as the Web, is a problem of growing interest [1, 2]. A significant step was taken by Downie [3], who applied standard text IR techniques to retrieve music by, initially, converting music to text format. Most research (including [3]) has, however, focused on plain music retrieval. To the best of our knowledge, there has been no attempt to model text and music jointly.

We propose a joint probabilistic model for documents with music and/or text. This model is simple, easily extensible, flexible and powerful. It allows users to query multimedia databases using text and/or music as input. It is well-suited for browsing applications as it organizes the documents into "soft" clusters. The document of highest probability in each cluster serves as a music thumbnail for automated music summarisation. The model allows one to query with an entire text document to automatically annotate the document with musical pieces. It can be used to automatically recommend or identify similar songs. Finally, it allows for the inclusion of different types of text, including website content, lyrics, and meta-data such as hyper-text links. The interested reader may further wish to consult [4], in which we discuss an application of our model to the problem of jointly

modelling music, as well as text and images.

## 2 Model specification

The training data consists of documents with text (lyrics or information about the song) and musical scores in GUIDO notation [5]. (GUIDO is a powerful language for representing musical scores in an HTML-like notation. MIDI files, plentiful on the World Wide Web, can be easily converted to this format.) We model the data with a Bayesian multi-modal mixture model. Words and scores are assumed to be conditionally independent given the mixture component label.

We model musical scores with first-order Markov chains, in which each state corresponds to a note, rest, or the start of a new voice. Notes' pitches are represented by the interval change (in semitones) from the previous note, rather than by absolute pitch, so that a score or query transposed to a different key will still have the same Markov chain. Rhythm is similarly represented as a scalar to the previous value. Rest states are represented similarly, save that pitch is not represented. See Figure 1 for an example.

Polyphonic scores are represented by chaining the beginning of a new voice to the end of a previous one. In order to ensure that the first note in each voice appears in both the row and column of the Markov transition matrix, a special "new voice" state with no interval or rhythm serves as a dummy state marking the beginning of a new voice. The first note of a voice has a distinguishing "first note" interval value and the first note or rest has a duration value of one.

[_*3/4 b&1*3/16 b1/16 c#2*11/16 b&1/16 a&1*3/16 b&1/16 f#1/2 ]

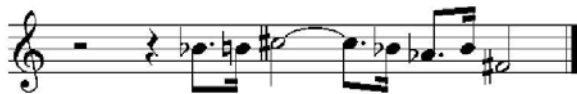

| $S$ | INTERVAL | DURATION |
|---|---|---|
| 0 | newvoice | 0 |
| 1 | rest | 1 |
| 2 | firstnote | 1/4 |
| 3 | +1 | 1/3 |
| 4 | +2 | 11 |
| 5 | -2 | 1/11 |
| 6 | -2 | 3 |
| 7 | +3 | 1/3 |
| 8 | -5 | 8 |

Figure 1: *Sample melody – the opening notes to "The Yellow Submarine" by The Beatles – in different notations. From top: GUIDO notation, standard musical notation (generated automatically from GUIDO notation), and as a series of states in a first-order Markov chain (also generated automatically from GUIDO notation).*

The Markov chain representation of a piece of music $k$ is then mapped to a sparse transition frequency table $\mathbf{M}_k$, where $\mathbf{M}_{i,j,k}$ denotes the number of times we observe the transition from state $i$ to state $j$ in document $k$. We use $\mathbf{M}_{k,0}$ to denote the initial state of the Markov chain. The associated text is modeled using a standard sparse term frequency vector $\mathbf{T}_k$, where $\mathbf{T}_{w,k}$ denotes the number of times word $w$ appears in document $k$. For notational simplicity, we group the music and text variable as follows: $\mathbf{X}_k \triangleq \{\mathbf{M}_k, \mathbf{T}_k\}$. In essence, this Markovian approach is akin to a text bigram model, save that the states are transitions between musical notes and rests rather than words.

Our multi-modal mixture model is as follows:

$$\mathbf{X}_k|\boldsymbol{\theta} \stackrel{iid}{\sim} \sum_{c=1}^{n_c} p(c) \left[ \prod_{j=1}^{n_s} p(j|c)^{\mathbb{I}_j(\mathbf{M}_{k,0})} \prod_{j=1}^{n_s} \prod_{i=1}^{n_s} p(j|i,c)^{M_{i,j,k}} \prod_{w=1}^{n_w} p(w|c)^{T_{w,k}} \right] \qquad (1)$$

where $\boldsymbol{\theta} \triangleq \{p(c), p(j|c), p(j|i,c), p(w|c)\}$ encompasses all the model parameters and where $\mathbb{I}_j(\mathbf{M}_{k,0}) = 1$ if the first entry of $\mathbf{M}_k$ belongs to state $j$ and is $0$ otherwise. The three-dimensional matrix $p(j|i,c)$ denotes the estimated probability of transitioning from state $i$ to state $j$ in cluster $c$, the matrix $p(j|c)$ denotes the initial probabilities of being in state $j$, given membership in cluster $c$. The vector $p(c)$ denotes the probability of each cluster. The matrix $p(w|c)$ denotes the probability of the word $w$ in cluster $c$. The mixture model is defined on the standard probability simplex $\{p(c) \geq 0 \text{ for all } c \text{ and } \sum_{c=1}^{n_c} p(c) = 1\}$. We introduce the latent allocation variables $z_k \in \{1, \ldots, n_c\}$ to indicate that a particular sequence $\mathbf{x}_k$ belongs to a specific cluster $c$. These indicator variables $\{z_k; k = 1, \ldots, n_x\}$ correspond to an i.i.d. sample from the distribution $p(z_k = c) = p(c)$.

This simple model is easy to extend. For browsing applications, we might prefer a hierarchical structure with levels $l$:

$$\mathbf{X}_k|\boldsymbol{\theta} \stackrel{iid}{\sim} \sum_{c=1}^{n_c} p(c) \sum_{l=1}^{n_l} p(l|c) p(\mathbf{M}_k|c,l) p(\mathbf{T}_k|c,l) \qquad (2)$$

This is still a multinomial model, but by applying appropriate parameter constraints we can produce a tree-like browsing structure [6]. It is also easy to formulate the model in terms of aspects and clusters as suggested in [7, 8].

## 2.1 Prior specification

We follow a hierarchical Bayesian strategy, where the unknown parameters $\boldsymbol{\theta}$ and the allocation variables $\mathbf{z}$ are regarded as being drawn from appropriate prior distributions. We acknowledge our uncertainty about the exact form of the prior by specifying it in terms of some unknown parameters (hyperparameters). The allocation variables $z_k$ are assumed to be drawn from a multinomial distribution, $z_k \sim \mathcal{M}_{n_c}(1; p(c))$. We place a conjugate Dirichlet prior on the mixing coefficients $p(c) \sim \mathcal{D}_{n_c}(\boldsymbol{\alpha})$. Similarly, we place Dirichlet prior distributions $\mathcal{D}_{n_j}(\boldsymbol{\beta})$ on each $p(j|c)$, $\mathcal{D}_{n_j}(\boldsymbol{\gamma})$ on each $p(j|i,c)$, $\mathcal{D}_{n_w}(\boldsymbol{\rho})$ on each $p(w|c)$, and assume that these priors are independent.

The posterior for the allocation variables will be required. It can be obtained easily using Bayes' rule:

$$p(c|k) \triangleq p(z_k = c|\boldsymbol{\theta}, \mathbf{X}_k) = \frac{p(\mathbf{X}_k|c, \boldsymbol{\theta}) p(c|\boldsymbol{\theta})}{p(\mathbf{X}_k|\boldsymbol{\theta})}$$

$$= \frac{p(c) \left( \prod_{j=1}^{n_s} p(j|c)^{\mathbb{I}_j(\mathbf{M}_{k,0})} \prod_{j=1}^{n_s} \prod_{i=1}^{n_s} p(j|i,c)^{M_{i,j,k}} \prod_{w=1}^{n_w} p(w|c)^{T_{w,c}} \right)}{\sum_{c'=1}^{n_c} p(c') \left( \prod_{j=1}^{n_s} p(j|c')^{\mathbb{I}_j(\mathbf{M}_{k,0})} \prod_{j=1}^{n_s} \prod_{i=1}^{n_s} p(j|i,c')^{M_{i,j,k}} \prod_{w=1}^{n_w} p(w|c')^{T_{w,c'}} \right)} \qquad (3)$$

## 3 Computation

The parameters of the mixture model cannot be computed analytically unless one knows the mixture indicator variables. We have to resort to numerical methods. One can implement a Gibbs sampler to compute the parameters and allocation variables. This is done by sampling the parameters from their Dirichlet posteriors and the allocation variables from their multinomial posterior. However, this algorithm is too computationally intensive for

the applications we have in mind. Instead we opt for expectation maximization (EM) algorithms to compute the maximum likelihood (ML) and *maximum a posteriori* (MAP) point estimates of the mixture model.

## 3.1 Maximum likelihood estimation with the EM algorithm

After initialization, the EM algorithm for ML estimation iterates between the following two steps:

**1. E step:** Compute the expectation of the complete log-likelihood with respect to the distribution of the allocation variables $\mathbf{Q}^{\text{ML}} = \mathbb{E}_{p(\mathbf{z}|\mathbf{M},\mathbf{T},\boldsymbol{\theta}^{(\text{old})})} \left[ \log p(\mathbf{z},\mathbf{M},\mathbf{T}|\boldsymbol{\theta}) \right]$, where $\boldsymbol{\theta}^{(\text{old})}$ represents the value of the parameters at the previous time step.

**2. M step:** Maximize over the parameters: $\boldsymbol{\theta}^{(\text{new})} = \arg\max_{\boldsymbol{\theta}} \mathbf{Q}^{\text{ML}}$

The $\mathbf{Q}^{\text{ML}}$ function expands to

$$\mathbf{Q}^{\text{ML}} = \sum_{k=1}^{n_x} \sum_{c=1}^{n_c} p(c|k) \log \left[ p(c) \prod_{j=1}^{n_s} p(j|c)^{\mathbb{I}_j(\mathbf{M}_{k,0})} \prod_{j=1}^{n_s} \prod_{i=1}^{n_s} p(j|i,c)^{M_{i,j,k}} \prod_{w=1}^{n_w} p(w|c)^{T_{w,k}} \right].$$

In the E step, we have to compute $p(c|k)$ using equation (3). The corresponding M step requires that we maximize $\mathbf{Q}^{\text{ML}}$ subject to the constraints that all probabilities for the parameters sum up to 1. This constrained maximization can be carried out by introducing Lagrange multipliers. The resulting parameter estimates are:

$$\widehat{p}(c) \quad = \quad \frac{1}{n_x} \sum_{k=1}^{n_x} p(c|k) \tag{5}$$

$$\widehat{p}(j|c) \quad = \quad \frac{\sum_{k=1}^{n_x} \mathbb{I}_j(M_{k,0}) p(c|k)}{\sum_{k=1}^{n_x} p(c|k)} \tag{6}$$

$$\widehat{p}(j|i,c) \quad = \quad \frac{\sum_{k=1}^{n_x} M_{i,j,k} p(c|k)}{\sum_{j=1}^{n_s} \sum_{k=1}^{n_x} M_{i,j,k} p(c|k)} \tag{7}$$

$$\widehat{p}(w|c) \quad = \quad \frac{\sum_{k=1}^{n_x} T_{w,k} p(c|k)}{\sum_{k=1}^{n_x} p(c|k)} \tag{8}$$

## 3.2 Maximum a posteriori estimation with the EM algorithm

The EM formulation for MAP estimation is straightforward. One simply has to augment the objective function in the M step, $\mathbf{Q}^{\text{ML}}$, by adding to it the log prior densities. That is, the MAP objective function is

$$\mathbf{Q}^{\text{MAP}} \quad = \quad \mathbb{E}_{p(\mathbf{z}|\mathbf{N},\boldsymbol{\theta}^{(\text{old})})} \left[ \log p(\mathbf{z},\mathbf{N},\boldsymbol{\theta}) \right] = \mathbf{Q}^{\text{ML}} + \log p(\boldsymbol{\theta})$$

The MAP parameter estimates are:

$$\widehat{p}(c) \quad = \quad \frac{\alpha_c - 1 + \sum_{k=1}^{n_x} p(c|k)}{\sum_{c'=1}^{n_c} \alpha_{c'} - n_c + n_x} \tag{9}$$

$$\widehat{p}(j|c) \quad = \quad \frac{\beta_{j,c} - 1 + \sum_{k=1}^{n_x} \mathbb{I}_j(M_{k,0}) p(c|k)}{\sum_{j'=1}^{n_s} \beta_{j',c} - n_s + \sum_{k=1}^{n_x} p(c|k)} \tag{10}$$

$$\widehat{p}(j|i,c) \quad = \quad \frac{\gamma_{i,j,c} - 1 + \sum_{k=1}^{n_x} M_{i,j,k} p(c|k)}{\sum_{j'=1}^{n_s} \gamma_{i,j',c} - n_s + \sum_{j=1}^{n_s} \sum_{k=1}^{n_x} M_{i,j,k} p(c|k)} \tag{11}$$

$$\widehat{p}(w|c) \quad = \quad \frac{\rho_{w,c} - 1 + \sum_{k=1}^{n_x} T_{w,k} p(c|k)}{\sum_{w'=1}^{n_w} \rho_{w',c} - n_w + \sum_{k=1}^{n_x} p(c|k)} \tag{12}$$

| CLUSTER | SONG | $p(c\|k)$ |
|---|---|---|
| 2 | *Moby – Porcelain* | 1 |
| 2 | *Nine Inch Nails – Terrible Lie* | 1 |
| 2 | *other – 'Addams Family' theme* | 1 |
| ⋮ | ⋮ | ⋮ |
| 4 | *J. S. Bach – Invention #1* | 1 |
| 4 | *J. S. Bach – Invention #8* | 1 |
| 4 | *J. S. Bach – Invention #15* | 1 |
| 4 | *The Beatles – Yellow Submarine* | 0.9975 |
| ⋮ | ⋮ | ⋮ |
| 6 | *other – 'Wheel of Fortune' theme* | 1 |
| ⋮ | ⋮ | ⋮ |
| 7 | *The Beatles – Taxman* | 1 |
| 7 | *The Beatles – Got to Get You Into My Life* | 0.7247 |
| 7 | *The Cure – Saturday Night* | 1 |
| ⋮ | ⋮ | ⋮ |
| 9 | *R.E.M – Man on the Moon* | 1 |
| 9 | *Soft Cell – Tainted Love* | 1 |
| 9 | *The Beatles – Got to Get You Into My Life* | 0.2753 |

Figure 2: *Representative probabilistic cluster allocations using MAP estimation.*

These expressions can also be derived by considering the posterior modes and by replacing the cluster indicator variable with its posterior estimate $p(c|k)$. This observation opens up room for various stochastic and deterministic ways of improving EM.

## 4  Experiments

To test the model with text and music, we clustered a database of musical scores with associated text documents. The database is composed of various types of musical scores – jazz, classical, television theme songs, and contemporary pop music – as well as associated text files. The scores are represented in GUIDO notation. The associated text files are a song's lyrics, where applicable, or textual commentary on the score for instrumental pieces, all of which were extracted from the World Wide Web.

The experimental database contains 100 scores, each with a single associated text document. There is nothing in the model, however, that requires this one-to-one association of text documents and scores – this was done solely for testing simplicity and efficiency. In a deployment such as the world wide web, one would routinely expect one-to-many or many-to-many mappings between the scores and text.

We carried out ML and MAP estimation with EM. The The Dirichlet hyper-parameters were set to $\alpha = 1, \beta = 10, \gamma = 10, \rho = 6$. The MAP approach resulted in sparser (regularised), more coherent clusters. Figure 2 shows some representative cluster probability assignments obtained with MAP estimation.

By and large, the MAP clusters are intuitive. The 15 pieces by J. S. Bach each have very high ($p > 0.999$) probabilities of membership in the same cluster. A few curious anomalies exist. The Beatles' song *The Yellow Submarine* is included in the same cluster as the Bach pieces, though all the other Beatles songs in the database are assigned to other clusters.

### 4.1 Demonstrating the utility of multi-modal queries

A major intended use of the text-score model is for searching documents on a combination of text and music.

Consider a hypothetical example, using our database: A music fan is struggling to recall a dimly-remembered song with a strong repeating single-pitch, dotted-eight-note/sixteenth-note bass line, and lyrics containing the words *come on, come on, get down*. A search on the text portion alone turns up four documents which contain the lyrics. A search on the notes alone returns seven documents which have matching transitions. But a combined search returns only the correct document (figure 3).

| QUERY | RETRIEVED SONGS |
|---|---|
| *come on, come on, get down* | |
| | *Erksine Hawkins – Tuxedo Junction* |
| | *Moby – Bodyrock* |
| | *Nine Inch Nails – Last* |
| | *Sherwood Schwartz – 'The Brady Bunch' theme song* |
|  | |
| | *The Beatles – Got to Get You Into My Life* |
| | *The Beatles – I'm Only Sleeping* |
| | *The Beatles – Yellow Submarine* |
| | *Moby – Bodyrock* |
| | *Moby – Porcelain* |
| | *Gary Portnoy – 'Cheers' theme song* |
| | *Rodgers & Hart – Blue Moon* |
| 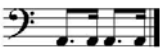 | |
| *come on, come on, get down* | |
| | *Moby – Bodyrock* |

Figure 3: *Examples of query matches, using only text, only musical notes, and both text and music. The combined query is more precise.*

### 4.2 Precision and recall

We evaluated our retrieval system with randomly generated queries. A query $\mathbf{Q}$ is composed of a random series of 1 to 5 note transitions, $\mathbf{Q}_m$ and 1 to 5 words, $\mathbf{Q}_t$. We then determine the actual number of matches $n$ in the database, where a match is defined as a song $\mathbf{X}_k$ such that all elements of $\mathbf{Q}_m$ and $\mathbf{Q}_t$ have a frequency of 1 or greater. In order to avoid skewing the results unduly, we reject any query that has $n < 5$ or $n > 20$.

To perform a query, we simply sample probabilistically without replacement from the clusters. The probability of sampling from each cluster, $p(c|\mathbf{Q})$, is computed using equation 3. If a cluster contains no items or later becomes empty, it is assigned a sampling probability of zero, and the probabilities of the remaining clusters are re-normalized.

In each iteration $i$, a cluster is selected, and the matching criteria are applied against each

piece of music that has been assigned to that cluster until a match is found. If no match is found, an arbitrary piece is selected. The selected piece is returned as the rank-$i^{th}$ result. Once all the matches have been returned, we compute the standard precision-recall curve [9], as shown in Figure 4.

Our querying method enjoys a high precision until recall is approximately $80\%$, and experiences a relatively modest deterioration of precision thereafter. By choosing clusters before

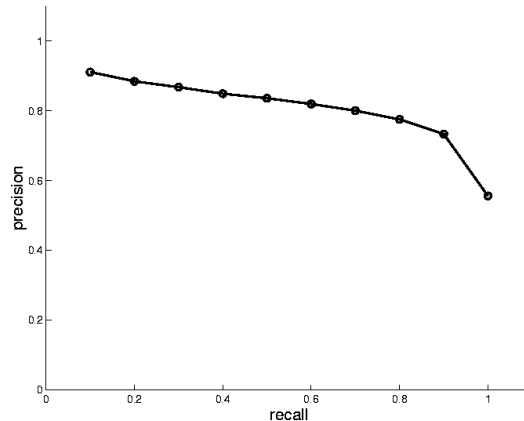

Figure 4: *Precision-recall curve showing average results, over 1000 randomly-generated queries, combining music and text matching criteria.*

matching, we overcome the polysemy problem. For example, *river banks* and *money banks* appear in separate clusters. We also deal with synonimy since *automobiles* and *cars* have high probability of belonging to the same clusters.

### 4.3 Association

The probabilistic nature of our approach allows us the flexibility to use our techniques and database for tasks beyond traditional querying. One of the more promising avenues of exploration is associating documents with each other probabilistically. This could be used, for example, to find suitable songs for web sites or presentations (matching on text), or for recommending songs similar to one a user enjoys (matching on scores).

Given an input document, $\mathbf{Q}$, we first cluster $\mathbf{Q}$ by finding the most likely cluster as determined by computing $\arg\max_c p(c|\mathbf{Q})$ (equation 3). Input documents containing text or music only can be clustered using only those components of the database. Input documents that combine text and music are clustered using all the data. We can then find the closest association by computing the distance from the input document to the other document vectors in the cluster using a similarity metric such as Euclidean distance, or cosine measures after carrying out latent semantic indexing [10]. A few selected examples of associations we found are shown in figure 5. The results are often reasonable, though unexpected behavior occasionally occurs.

## 5   Conclusions

We feel that the probabilistic approach to querying on music and text presented here is powerful, flexible, and novel, and suggests many interesting areas of future research. In the future, we should be able to incorporate audio by extracting suitable features from the

| INPUT | CLOSEST MATCH |
|---|---|
| *J. S. Bach – Toccata and Fugue in D Minor* (score) | *J. S. Bach – Invention #5* |
| *Nine Inch Nails – Closer* (score & lyrics) | *Nine Inch Nails – I Do Not Want This* |
| *T. S. Eliot – The Waste Land* (text poem) | *The Cure – One Hundred Years* |

Figure 5: *The results of associating songs in the database with other text and/or musical input. The input is clustered probabilistically and then associated with the existing song that has the least Euclidean distance in that cluster. The association of* The Wasteland *with* The Cure's *thematically similar* One Hundred Years *is likely due to the high co-occurance of relatively uncommon words such as* water*,* death*, and* year(s).

signals. This will permit querying by singing, humming, or via recorded music. There are a number of ways of combining our method with images [6, 4], opening up room for novel applications in multimedia [11].

## Acknowledgments

We would like to thank Kobus Barnard, J. Stephen Downie, Holger Hoos and Peter Carbonetto for their advice and expertise in preparing this paper.

## References

[1] D Huron and B Aarden. Cognitive issues and approaches in music information retrieval. In S Downie and D Byrd, editors, *Music Information Retrieval*. 2002.

[2] J Pickens. A comparison of language modeling and probabilistic text information retrieval approaches to monophonic music retrieval. In *International Symposium on Music Information Retrieval*, 2000.

[3] J S Downie. *Evaluating a Simple Approach to Music Information Retrieval: Conceiving Melodic N-Grams as Text*. PhD thesis, University of Western Ontario, 1999.

[4] E Brochu, N de Freitas, and K Bao. The sound of an album cover: Probabilistic multimedia and IR. In C M Bishop and B J Frey, editors, *Ninth International Workshop on Artificial Intelligence and Statistics*, Key West, Florida, 2003. To appear.

[5] H H Hoos, K A Hamel, K Renz, and J Kilian. Representing score-level music using the GUIDO music-notation format. *Computing in Musicology*, 12, 2001.

[6] K Barnard and D Forsyth. Learning the semantics of words and pictures. In *International Conference on Computer Vision*, volume 2, pages 408– 415, 2001.

[7] T Hofmann. Probabilistic latent semantic analysis. In *Uncertainty in Artificial Intelligence*, 1999.

[8] D M Blei, A Y Ng, and M I Jordan. Latent Dirichlet allocation. In T G Dietterich, S Becker, and Z Ghahramani, editors, *Advances in Neural Information Processing Systems 14*, Cambridge, MA, 2002. MIT Press.

[9] R Baeza-Yates and B Ribeiro-Neto. *Modern Information Retrieval*. Addison-Wesley, 1999.

[10] S Deerwester, S T Dumais, G W Furnas, T K Landauer, and R Harshman. Indexing by latent semantic indexing. *Journal of the American Society for Information Science*, 41(6):391– 407, 1990.

[11] P Duygulu, K Barnard, N de Freitas, and D Forsyth. Object recognition as machine translation: Learning a lexicon for a fixed image vocabulary. In *ECCV*, 2002.
